# A Predictive Switching Model of Cerebellar Movement Control

**Andrew G. Barto**
**Jay T. Buckingham**
Department of Computer Science
University of Massachusetts
Amherst, MA 01003-4610
barto@cs.umass.edu

**James C. Houk**
Department of Physiology
Northwestern University Medical School
303 East Chicago Ave
Chicago, Illinois 60611-3008
houk@acns.nwu.edu

## Abstract

We present a hypothesis about how the cerebellum could participate in regulating movement in the presence of significant feedback delays without resorting to a forward model of the motor plant. We show how a simplified cerebellar model can learn to control endpoint positioning of a nonlinear spring–mass system with realistic delays in both afferent and efferent pathways. The model's operation involves prediction, but instead of predicting sensory input, it directly regulates movement by reacting in an anticipatory fashion to input patterns that include delayed sensory feedback.

## 1  INTRODUCTION

The existence of significant delays in sensorimotor feedback pathways has led several researchers to suggest that the cerebellum might function as a forward model of the motor plant in order to predict the sensory consequences of motor commands before actual feedback is available; e.g., (Ito, 1984; Keeler, 1990; Miall *et al.*, 1993). While we agree that there are many potential roles for forward models in motor control systems, as discussed, e.g., in (Wolpert *et al.*, 1995), we present a hypothesis about how the cerebellum could participate in regulating movement in the presence of significant feedback delays without resorting to a forward model. We show how a very simplified version of the adjustable pattern generator (APG) model being developed by Houk and colleagues (Berthier *et al.*, 1993; Houk *et al.*, 1995) can learn to control endpoint positioning of a nonlinear spring–mass system with significant delays in both afferent and efferent pathways. Although much simpler than a multilink dynamic arm, control of this spring–mass system involves some of the challenges critical in the control of a more realistic motor system and serves to illustrate the principles we propose. Preliminary results appear in (Buckingham *et al.*, 1995).

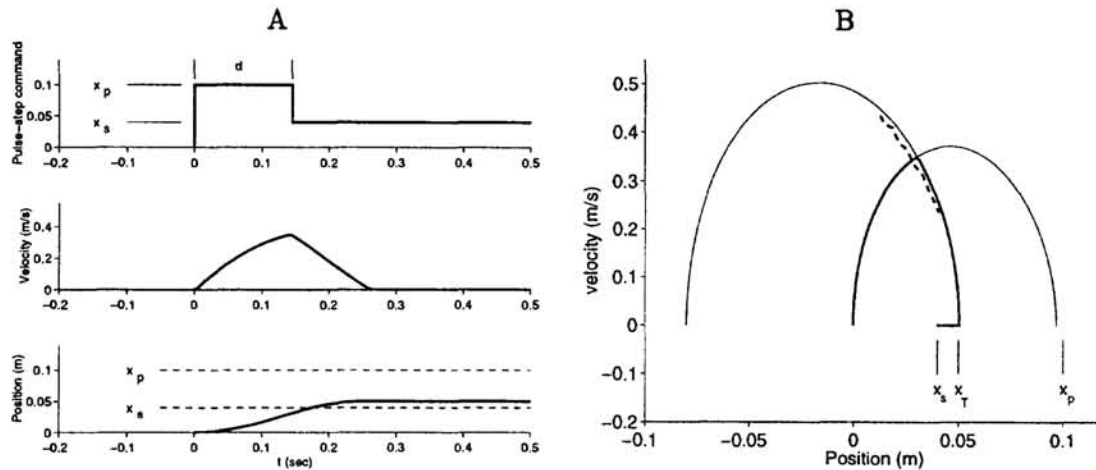

Figure 1: Pulse–step control of a movement from initial position $x_0 = 0$ to target endpoint position $x_T = .05$. Panel A: Top—The pulse–step command. Middle—Velocity as a function of time. Bottom—Position as a function of time. Panel B: Switching curve. The dashed line plots states of the spring–mass system at which the command should switch from pulse to step so that the mass will stick at the endpoint $x_T = .05$ starting from different initial states. The bold line shows the phase–plane trajectory of the movement shown in Panel A.

## 2   NONLINEAR VISCOSITY

An important aspect of the model is that the plant being contolled has a form of nonlinear viscosity, brought about in animals through a combination of muscle and spinal reflex properties. To illustrate this, we use a nonlinear spring–mass model based on studies of human wrist movement (Wu *et al.*, 1990):

$$m\ddot{x} + b\dot{x}^{\frac{1}{5}} + k(x - x_{eq}) = 0, \qquad (1)$$

where $x$ is the position (in meters) of an object of mass $m$ (kg) attached to the spring, $x_{eq}$ is the resting, or equilibrium, position, $b$ is a damping coefficient, and $k$ is the spring's stiffness. Setting $m = 1$, $b = 4$, and $k = 60$ produces trajectories that are qualitatively similar to those observed in human wrist movement (Wu *et al.*, 1990).

This one–fifth power law viscosity gives the system the potential to produce fast movements that terminate with little or no oscillation. However, the principle of setting the equilibrium position to the desired movement endpoint does not work in practice because the system tends to "stick" at non–equilibrium positions, thereafter drifting extremely slowly toward the equilibrium position, $x_{eq}$. We call the position at which the mass sticks (which we define as the position at which its absolute velocity falls and remains below .005m/s) the *endpoint* of a movement, denoted $x_e$. Thus, endpoint control of this system is not entirely straightforward. The approach taken by our model is to switch the value of the control signal, $x_{eq}$, at a precisely–placed point during a movement. This is similar to virtual trajectory control, except that here the commanded equilibrium position need not equal the desired endpoint either before or after the switch.

Panel A of Fig. 1 shows an example of this type of control. The objective is to move the mass from an initial position $x_0 = 0$ to a target endpoint $x_T = .05$. The control signal is the pulse–step shown in the top graph, where $x_p = .1$ and $x_s = .04$

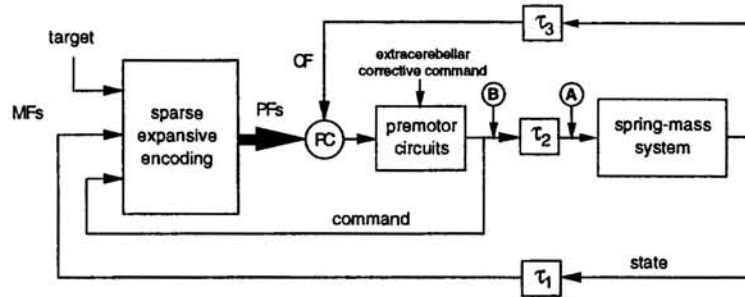

Figure 2: The simplified model. PC, Purkinje cell; MFs, mossy fibers; PFs, parallel fibers; CF, climbing fiber. The labels A and B mark places in the feedback loop to which we refer in discussing the model's behavior.

respectively denote the pulse and step values, and $d$ denotes the pulse duration. The mass sticks near the target endpoint $x_T = .05$, which is different from both equilibrium positions. If the switch had occurred sooner (later), the mass would have undershot (overshot) the target endpoint.

The bold trajectory in Panel B of Fig. 1 is the phase–plane portrait of this movement. During its initial phase, the state follows the trajectory that would eventually lead to equilibrium position $x_p$. When the pulse ends, the state switches to the trajectory that would eventually lead to equilibrium position $x_s$, which allows a rapid approach to the target endpoint $x_T = .05$, where the mass sticks before reaching $x_s$. The dashed line plots pairs of positions and velocities at which the switch should occur so that movements starting from different initial states will reach the endpoint $x_T = .05$. This *switching curve* has to vary as a function of the target endpoint.

## 3  THE MODEL'S ARCHITECTURE

The simplified model (Fig. 2) consists of a unit representing a Purkinje cell (PC) whose input is derived from a sparse expansive encoding of mossy fiber (MF) input representing the target position, $x_T$, which remains fixed throughout a movement, delayed information about the state of the spring–mass system, and the current motor command, $x_{eq}$.[1] Patterns of MF activity are recoded to form sparse activity patterns over a large number (here 8000) of binary parallel fibers (PFs) which synapse upon the PC unit, along the lines suggested by Marr (Marr, 1969) and the CMAC model of Albus (Albus, 1971). While some liberties have been taken with this representation, the delay distributions are within the range observed for the intermediate cerebellum of the monkey (Van Kan *et al.*, 1993).

Also as in Marr and Albus, the PC unit is trained by a signal representing the activity of a climbing fiber (CF), whose response properties are described below. Occasional corrective commands, also discussed below, are assumed to be generated

by an extracerebellar system. The PC's output determines the motor command through a simple transformation. The model includes an efferent and CF delays, both equal to 20msec ($\tau_2$ and $\tau_3$, respectively, in Fig. 2). These delays are also within the physiological range for these pathways (Gellman *et al.*, 1983). How this model is related to the full APG model and its justification in terms of the anatomy and physiology of the cerebellum and premotor circuits are discussed extensively elsewhere (Berthier *et al.*, 1993; Houk *et al.*, 1995).

The PC unit is a linear threshold unit with hysteresis. Let $s(t) = \sum_i w_i(t)\phi_i(t)$, where $\phi_i(t)$ denotes the activity of PF $i$ at time $t$ and $w_i(t)$ is the weight at time step $t$ of the synapse by which PF $i$ influences the PC unit. The output of the PC unit at time $t$, denoted $y(t)$, is the PC's activity state, high or low, at time $t$, which represents a high or a low frequency of simple spike activity. PC activation depends on two thresholds: $\theta_{high}$ and $\theta_{low} < \theta_{high}$. The activity state switches from low to high when $s(t) > \theta_{high}$, and it switches from high to low when $s(t) < \theta_{low}$. If $\theta_{high} = \theta_{low}$, the PC unit is the usual linear threshold unit. Although hysteresis is not strictly necessary for the control task we present here, it accelerates learning: A PC can more easily learn when to switch states than it can learn to maintain the correct output on a moment–to–moment basis. The bistability of this PC unit is a simplified representation of multistability that could be produced by dendritic zones of hysteresis arising from ionic mechanisms (Houk *et al.*, 1995).

Because PC activity inhibits premotor circuits, PC state low corresponds to the pulse phase of the motor command, which sets a "far" equilibrium position, $x_p$; PC state high corresponds to the step phase, which sets a "near" equilibrium position, $x_s$. Thus, the pulse ends when the PC state switches from low to high. Because the precise switching point determines where the mass sticks, this single binary PC can bring the mass to any target endpoint in a considerable range by switching state at the right moment during a movement.

## 4   LEARNING

Learning is based on the idea that corrective movements following inaccurate movements provide training information by triggering CF responses. These responses are presumed to be proprioceptively triggered by the onset of a corrective movement, being suppressed during the movement itself. Corrective movements can be generated when a cerebellar module generates an additional pulse phase of the motor command, or through the action of a system other than the cerebellum. The second, extracerebellar, source of corrective movements only needs to operate when small corrections are needed.

The learning mechanism has to adjust the PC weights, $w_i$, so that the PC switches state at the correct moment during a movement. This is difficult because training information is significantly delayed due to the combined effects of movement duration and delays in the relevant feedback pathways. The relevant PC activity is completed well before a corrective movement triggers a CF response. To learn under these conditions, the learning mechanism needs to modify synaptic actions that occurred prior to the CF's discharge. The APG model adopts Klopf's (Klopf, 1982) idea of a synaptic "eligibility trace" whereby appropriate synaptic activity sets up a synaptically–local memory trace that renders the synapse "eligible" for modification if and when the appropriate training information arrives within a short time period.

The learning rule has two components: one implments a form of long–term depression (LTD); the other implements a much weaker form of long–term potentiation

(LTP). It works as follows. Whenever the CF fires ($c(t) = 1$), the weights of all the *eligible* synapses decrease. A synapse is eligible if its presynaptic parallel fiber was active in the past when the PC switched from low to high, with the degree of eligibility decreasing with the time since that state switch. This makes the PC less likely to switch to high in future situations represented by patterns of PF activity similar to the pattern present when the eligibility–initiating switch occurred. This has the effect of increasing the duration of the PC pause, which increases the duration of the pulse phase of the motor command. Superimposed on weight decreases are much smaller weight increases that occur for any synapse whose presynaptic PF is active when the PC switches from low to high, irrespective of CF activity. This makes the PC more likely to switch to high under similar circumstances in the future, which decreases the duration of the pulse phase of the movement command.

To define this mathematically, let $\eta(t)$ detect when the PC's activity state switches from low to high: $\eta(t) = 0$ unless $y(t - 1) = low$ and $y(t) = high$, in which case $\eta(t) = 1$. The eligibility trace for synapse $i$ at time step $t$, denoted $e_i(t)$, is set to 1 whenever $\eta(t) = 1$ and thereafter decays geometrically toward zero until it is reset to 1 when $\eta$ is again set to 1 by another upward switch of PC activity level. Then the learning rule is given for $t = 1, 2, \ldots,$ by:

$$\Delta w_i(t) = -\alpha c(t)e_i(t)+\beta\eta(t)\phi_i(t),$$

where $\alpha$ and $\beta$, with $\alpha \gg \beta$, are positive parameters respectively determining the rate of LTD and LTP. See (Houk et al., 1995) for a discussion of this learning rule in light of physiological data and cellular mechanisms.

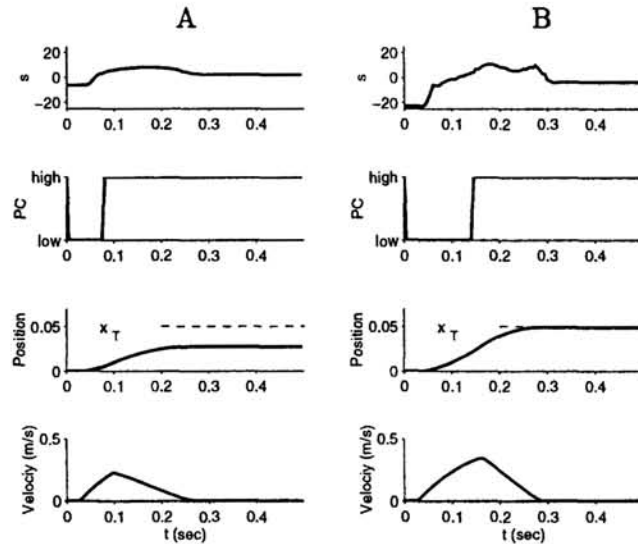

Figure 3: Model behavior. Panel A: early in learning; Panel B: late in learning. Assume that at time step 0, $x_T$ has just been switched from 0 to .05. Shown are the time courses of the PC's weighted sum, $s$, activation state, $y$, and the position and velocity of the mass.

## 5  SIMULATIONS

We performed a number of simulations of the simplified APG model learning to control the nonlinear spring–mass system. We trained each version of the model to move the mass from initial positions selected randomly from the interval $[-.02, .02]$ to a target position randomly set to .03, .04, or .05. We set the pulse height, $x_p$, and the step height, $x_s$, to .1 and .04 respectively. Each simulation consisted of a series of trial movements. The parameters of the learning rule, which were not optimized, were $\alpha = .0004$ and $\beta = .00004$. Eligibility traces decayed 1% per time step.

Figure 3 shows time courses of relevant variables at different stages in learning to move to target endpoint $x_T = .05$ from initial position $x_0 = 0$. Early in learning (Panel A), the PC has learned to switch to low at the beginning of the trial but

switches back to high too soon, which causes the mass to undershoot the target. Because of this undershoot, the CF fires at the end of the movement due to a final very small corrective movement generated by an extracerebellar system. The mass sticks at $x_e = .027$. Late in learning (Panel B), the mass sticks at $x_e = .049$, and the CF does not fire. Note that to accomplish this, the PC state has to switch to high well before (about 150ms) the endpoint is reached.

Figure 4 shows three representations of the switching curve learned by a version of the model for target $x_T = .05$. As an aid to understanding the model's behavior, all the proprioceptive signals in this version of the model had the same delay of 30ms ($\tau_1$ in Fig. 2) instead of the more realistic distribution of delays described above. Hence the total loop delay ($\tau_1 + \tau_2$) was 50ms. The curve labeled "spring switch", which closely coincides with the optimal switching curve (also shown), plots states that the spring–mass system passes through when the command input to the spring switches. In other words, this is the switching curve as seen from the point marked A in Fig. 2. That this coincides with the optimal switching curve shows that the model learned to behave correctly. The movement trajectory crosses this curve about 150ms before the movement ends.

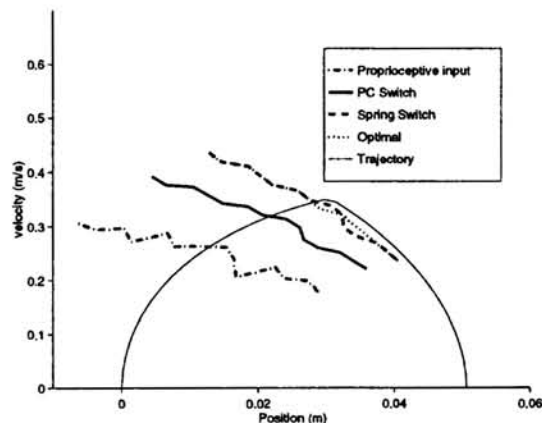

Figure 4: Phase–plane portraits of switching curves implemented by the model after learning. Four switching curves and one movement trajectory are shown. See text for explanation.

The curve labeled "PC switch", on the other hand, plots states that the spring–mass system passes through when the PC unit switches state: it is the switching curve as seen from the point marked B in Fig. 2 (assuming the expansive encoding involves no delay). The state of the spring–mass system crosses this curve 20ms before it reaches the "spring switch" curve. One can see, therefore, that the PC unit learned to switch its activity state 20ms before the motor command must switch state at the spring itself, appropriately compensating for the 20ms latency of the efferent pathway.

We can also ask what is the state of the spring–mass system that the PC actually "sees", via proprioceptive signals, when it has to switch state. When the PC has to switch states, that is, when the spring–mass state reaches switching curve "PC switch", the PC is actually receiving via its PF input a description of the system state that occurred a significant time earlier ($\tau_1 = 30$ms in Fig. 2). Switching curve "proprioceptive input" in Fig. 4 is the locus of system states that the PC is sensing when it has to switch. The PC has learned to do this by learning, on the basis of delayed CF training information, to switch when it sees PF patterns that code spring–mass states that lie on curve "proprioceptive input".

## 6  DISCUSSION

The model we have presented is most closely related to adaptive control methods known as direct predictive adaptive controllers (Goodwin & Sin, 1984). Feedback delays pose no particular difficulties despite the fact that no use is made of a forward model of the motor plant. Instead of producing predictions of proprioceptive

feedback, the model uses its predictive capabilities to directly produce appropriately timed motor commands. Although the nonlinear viscosity of the spring–mass system renders linear control principles inapplicable, it actually makes the control problem easier for an appropriate controller. Fast movements can be performed with little or no oscillation. We believe that similar nonlinearities in actual motor plants have significant implications for motor control. A critical feature of this model's learning mechanism is its use of eligibility traces to bridge the temporal gap between a PC's activity and the consequences of this activity on the movement endpoint. Cellular studies are needed to explore this important issue. Although nothing in the present paper suggests how this might extend to more complex control problems, one of the objectives of the full APG model is to explore how the collective behavior of multiple APG modules might accomplish more complex control.

## Acknowledgements

This work was supported by NIH 1-50 MH 48185-04.

## Footnotes

[1]In this model, 256 Gaussian radial basis function (RBF) units represent the target position, 400 RBF units represent the position of the mass (i.e., the length of the spring), with centers distributed uniformly across an appropriate range of positions and with delays distributed according to a Gaussian of mean 15msec and standard deviation 6msec. This distribution is truncated so that the minimum delay is 5msec. This delay distribution is represented by $\tau_1$ in Fig. 2. Another 400 RBF units similarly represent mass velocity. An additional 4 MF inputs are efference copy signals that simply copy the current motor command.

## References

Albus, JS (1971). A theory of cerebellar function. *Mathematical Biosciences*, **10**, 25–61.

Berthier, NE, Singh, SP, Barto, AG, & Houk, JC (1993). Distributed representations of limb motor programs in arrays of adjustable pattern generators. *Cognitive Neuroscience*, **5**, 56–78.

Buckingham, JT, Barto, AG, & Houk, JC (1995). Adaptive predictive control with a cerebellar model. In: *Proceedings of the 1995 World Congress on Neural Networks*, I-373–I-380.

Gellman, R, Gibson, AR, & Houk, JC (1983). Somatosensory properties of the inferior olive of the cat. *J. Comp. Neurology*, **215**, 228–243.

Goodwin, GC & Sin, KS (1984). *Adaptive Filtering Prediction and Control.* Englewood Cliffs, N.J.: Prentice-Hall.

Houk, JC, Buckingham, JT, & Barto, AG (1995). Models of the cerebellum and motor learning. *Brain and Behavioral Sciences*, in press.

Ito, M (1984). *The Cerebellum and Neural Control.* New York: Raven Press.

Keeler, JD (1990). A dynamical system view of cerebellar function. *Physica D*, **42**, 396–410.

Klopf, AH (1982). *The Hedonistic Neuron: A Theory of Memory, Learning, and Intelligence.* Washington, D.C.: Hemishere.

Marr, D (1969). A theory of cerebellar cortex. *J. Physiol. London*, **202**, 437–470.

Miall, RC, Weir, DJ, Wolpert, DM, & Stein, JF (1993). Is the cerebellum a smith predictor? *Journal of Motor Behavior*, **25**, 203–216.

Van Kan, PLE, Gibson, AR, & Houk, JC (1993). Movement-related inputs to intermediate cerebellum of the monkey. *Journal of Physiology*, **69**, 74–94.

Wolpert, DM, Ghahramani, Z, & Jordan, MI (1995). Foreward dynamic models in human motor control: Psychophysical evidence. In: *Advances in Neural Information Processing Systems 7*, (G Tesauro, DS Touretzky, & TK Leen, eds) , Cambridge, MA: MIT Press.

Wu, CH, Houk, JC, Young, KY, & Miller, LE (1990). Nonlinear damping of limb motion. In: *Multiple Muscle Systems: Biomechanics and Movement Organization*, (J Winters & S Woo, eds). New York: Springer–Verlag.
